# Nonparametric Representation of Policies and Value Functions: A Trajectory-Based Approach

**Christopher G. Atkeson**[*]
Robotics Institute and HCII
Carnegie Mellon University
Pittsburgh, PA 15213, USA
`cga@cmu.edu`

**Jun Morimoto**
ATR Human Information Science Laboratories, Dept. 3
Keihanna Science City
Kyoto 619-0288, Japan
`xmorimo@atr.co.jp`

## Abstract

A longstanding goal of reinforcement learning is to develop nonparametric representations of policies and value functions that support rapid learning without suffering from interference or the curse of dimensionality. We have developed a trajectory-based approach, in which policies and value functions are represented nonparametrically along trajectories. These trajectories, policies, and value functions are updated as the value function becomes more accurate or as a model of the task is updated. We have applied this approach to periodic tasks such as hopping and walking, which required handling discount factors and discontinuities in the task dynamics, and using function approximation to represent value functions at discontinuities. We also describe extensions of the approach to make the policies more robust to modeling error and sensor noise.

## 1  Introduction

The widespread application of reinforcement learning is hindered by excessive cost in terms of one or more of representational resources, computation time, or amount of training data. The goal of our research program is to minimize these costs. We reduce the amount of training data needed by learning models, and using a DYNA-like approach to do mental practice in addition to actually attempting a task [1, 2]. This paper addresses concerns about computation time and representational resources. We reduce the computation time required by using more powerful updates that update first and second derivatives of value functions and first derivatives of policies, in addition to updating value function and policy values at particular points [3, 4, 5]. We reduce the representational resources needed by representing value functions and policies along carefully chosen trajectories. This non-parametric representation is well suited to the task of representing and updating value functions, providing additional representational power as needed and avoiding interference.

This paper explores how the approach can be extended to periodic tasks such as hopping and walking. Previous work has explored how to apply an early version of this approach to tasks with an explicit goal state [3, 6] and how to simultaneously learn a model and

---

[*]also affiliated with the ATR Human Information Science Laboratories, Dept. 3

use this approach to compute a policy and value function [6]. Handling periodic tasks required accommodating discount factors and discontinuities in the task dynamics, and using function approximation to represent value functions at discontinuities.

## 2   What is the approach?

**Represent value functions and policies along trajectories.** Our first key idea for creating a more global policy is to coordinate many trajectories, similar to using the method of characteristics to solve a partial differential equation. A more global value function is created by combining value functions for the trajectories. As long as the value functions are consistent between trajectories, and cover the appropriate space, the global value function created will be correct. This representation supports accurate updating since any updates must occur along densely represented optimized trajectories, and an adaptive resolution representation that allocates resources to where optimal trajectories tend to go.

**Segment trajectories at discontinuities.** A second key idea is to segment the trajectories at discontinuities of the system dynamics, to reduce the amount of discontinuity in the value function within each segment, so our extrapolation operations are correct more often. We assume smooth dynamics and criteria, so that first and second derivatives exist. Unfortunately, in periodic tasks such as hopping or walking the dynamics changes discontinuously as feet touch and leave the ground. The locations in state space at which this happens can be localized to lower dimensional surfaces that separate regions of smooth dynamics. For periodic tasks we apply our approach along trajectory segments which end whenever a dynamics (or criterion) discontinuity is reached. We also search for value function discontinuities not collocated with dynamics or criterion discontinuities. We can use all the trajectory segments that start at the discontinuity and continue through the next region to provide estimates of the value function at the other side of the discontinuity.

**Use function approximation to represent value function at discontinuities.** We use locally weighted regression (LWR) to construct value functions at discontinuities [7].

**Update first and second derivatives of the value function as well as first derivatives of the policy (control gains for a linear controller) along the trajectory.** We can think of this as updating the first few terms of local Taylor series models of the global value and policy functions. This non-parametric representation is well suited to the task of representing and updating value functions, providing additional representational power as needed and avoiding interference.

We will derive the update rules. Because we are interested in periodic tasks, we must introduce a discount factor into Bellman's equation, so value functions remain finite. Consider a system with dynamics $\mathbf{x}_{k+1} = \mathbf{f}(\mathbf{x}_k, \mathbf{u}_k)$ and a one step cost function $L(\mathbf{x}_k, \mathbf{u}_k)$, where $\mathbf{x}$ is the state of the system and $\mathbf{u}$ is a vector of actions or controls. The subscript $k$ serves as a time index, but will be dropped in the equations that follow in cases where all time indices are the same or are equal to $k$.

A goal of reinforcement learning and optimal control is to find a policy that minimizes the total cost, which is the sum of the costs for each time step. One approach to doing this is to construct an optimal value function, $V(\mathbf{x})$. The value of this value function at a state $\mathbf{x}$ is the sum of all future costs, given that the system started in state $\mathbf{x}$ and followed the optimal policy (chose optimal actions at each time step as a function of the state). A local planner or controller can choose globally optimal actions if it knew the future cost of each action. This cost is simply the sum of the cost of taking the action right now and the discounted future cost of the state that the action leads to, which is given by the value function. Thus, the optimal action is given by: $\mathbf{u} = \arg\min_{\mathbf{u}} \left( L(\mathbf{x}, \mathbf{u}) + \lambda V(\mathbf{f}(\mathbf{x}, \mathbf{u})) \right)$ where $\lambda$ is the discount factor.

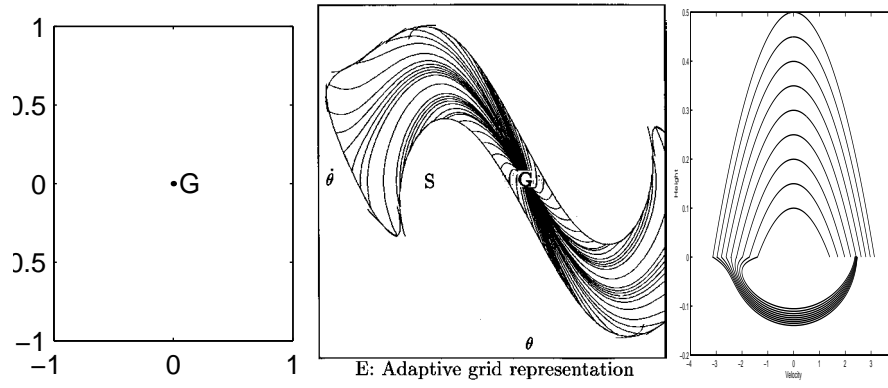

Figure 1: Example trajectories where the value function and policy are explicitly represented for a regulator task at goal state G (left), a task with a point goal state G (middle), and a periodic task (right).

Suppose at a point $(\mathbf{x}_p, \mathbf{u}_p)$ we have 1) a local second order Taylor series approximation of the optimal value function: $V(\mathbf{x}) \approx V_0 + V_x\tilde{\mathbf{x}} + \frac{1}{2}\tilde{\mathbf{x}}^T V_{xx}\tilde{\mathbf{x}}$ where $\tilde{\mathbf{x}} = \mathbf{x} - \mathbf{x}_p$. 2) a local second order Taylor series approximation of the dynamics, which can be learned using local models of the plant ($\mathbf{f}_x$ and $\mathbf{f}_u$ correspond to the usual $\mathbf{A}$ and $\mathbf{B}$ of the linear plant model used in linear quadratic regulator (LQR) design): $\mathbf{x}_{k+1} = \mathbf{f}(\mathbf{x}, \mathbf{u}) \approx \mathbf{f}_0 + \mathbf{f}_x\tilde{\mathbf{x}} + \mathbf{f}_u\tilde{\mathbf{u}} + \frac{1}{2}\tilde{\mathbf{x}}^T\mathbf{f}_{xx}\tilde{\mathbf{x}} + \tilde{\mathbf{x}}^T\mathbf{f}_{xu}\tilde{\mathbf{u}} + \frac{1}{2}\tilde{\mathbf{u}}^T\mathbf{f}_{uu}\tilde{\mathbf{u}}$ where $\tilde{\mathbf{u}} = \mathbf{u} - \mathbf{u}_p$, and 3) a local second order Taylor series approximation of the one step cost, which is often known analytically for human specified criteria ($L_{xx}$ and $L_{uu}$ correspond to the usual $\mathbf{Q}$ and $\mathbf{R}$ of LQR design): $L(\mathbf{x}, \mathbf{u}) \approx L_0 + L_x\tilde{\mathbf{x}} + L_u\tilde{\mathbf{u}} + \frac{1}{2}\tilde{\mathbf{x}}^T L_{xx}\tilde{\mathbf{x}} + \tilde{\mathbf{x}}^T L_{xu}\tilde{\mathbf{u}} + \frac{1}{2}\tilde{\mathbf{u}}^T L_{uu}\tilde{\mathbf{u}}$

Given a trajectory, one can integrate the value function and its first and second spatial derivatives backwards in time to compute an improved value function and policy. The backward sweep takes the following form (in discrete time):

$$Q_x = L_x + \lambda V_x\mathbf{f}_x; Q_u = L_u + \lambda V_x\mathbf{f}_u \tag{1}$$

$$Q_{xx} = \lambda\mathbf{f}_x^T V_{xx}\mathbf{f}_x + \lambda V_x\mathbf{f}_{xx} + L_{xx}; Q_{ux} = \lambda\mathbf{f}_u^T V_{xx}\mathbf{f}_x + \lambda V_x\mathbf{f}_{ux} + L_{ux} \tag{2}$$

$$Q_{uu} = \lambda\mathbf{f}_u^T V_{xx}\mathbf{f}_u + \lambda V_x\mathbf{f}_{uu} + L_{uu} \tag{3}$$

$$\Delta\mathbf{u} = Q_{uu}^{-1}Q_u; \mathbf{K} = Q_{uu}^{-1}Q_{ux} \tag{4}$$

$$V_{x_{k-1}} = Q_x - Q_u\mathbf{K}; V_{xx_{k-1}} = Q_{xx} - Q_{xu}\mathbf{K} \tag{5}$$

After the backward sweep, forward integration can be used to update the trajectory itself: $\mathbf{u}_{new} = \mathbf{u} - \Delta\mathbf{u} - \mathbf{K}(\mathbf{x}_{new} - \mathbf{x})$

In order to use this approach we have to assume smooth dynamics and criteria, so that first and second derivatives exist. Unfortunately, in periodic tasks such as hopping or walking the dynamics changes discontinuously as feet touch and leave the ground. The locations in state space at which this happens can be localized to lower dimensional surfaces that separate regions of smooth dynamics. For periodic tasks we apply our approach along trajectory segments which end whenever a dynamics (or criterion) discontinuity is reached. We can use all the trajectory segments that start at the discontinuity and continue through the next region to provide estimates of the value function at the other side of the discontinuity.

Figure 1 shows our approach applied to several types of problems. On the left we see that a task that requires steady state control about a goal point (a regulator task) can be solved with a single trivial trajectory that starts and ends at the goal and provides a value function and constant linear policy $\tilde{\mathbf{u}} = \mathbf{K}\tilde{\mathbf{x}}$ in the vicinity of the goal.

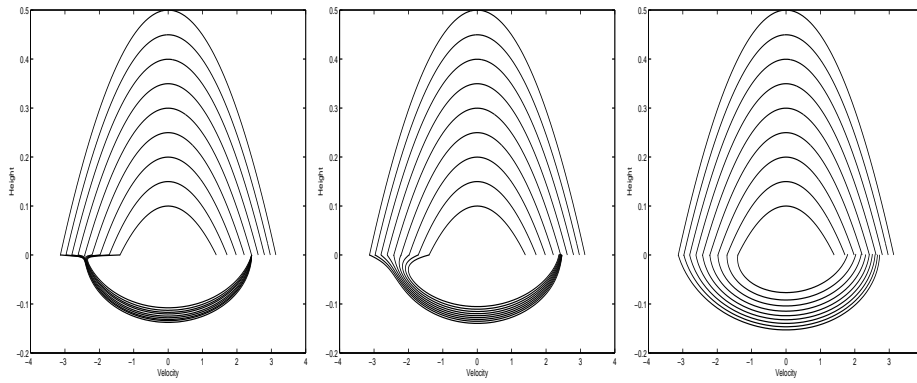

Figure 2: The optimal hopper controller with a range of penalties on $\mathbf{u}$ usage ($R = 1, 100, 10000$). $C(\mathbf{x}, \mathbf{u}) = 4 * (Energy - E_d)^2 + R * u^2$

The middle figure of Figure 1 shows the trajectories used to compute the value function for a swing up problem [3]. In this problem the goal requires regulation about the state where the pendulum is inverted and in an unstable equilibrium. However, the nonlinearities of the problem limit the region of applicability of a linear policy, and non-trivial trajectories have to be created to cover a larger region. In this case the region where the value function is less than a target value is filled with trajectories. The neighboring trajectories have consistent value functions and thus the globally optimal value function and policy is found in the explored region [3].

The right figure of Figure 1 shows the trajectories used to compute the value function for a periodic problem, control of vertical hopping in a hopping robot. In this problem, there is no goal state, but a desired hopping height is specified. This problem has been extensively studied in the robotics literature [8] from the point of view of how to manually design a nonlinear controller with a large stability region. We note that optimal control provides a methodology to design nonlinear controllers with large stability regions and also good performance in terms of explicitly specified criteria. We describe later how to also make these controller designs more robust.

In this figure the vertical axis corresponds to the height of the hopper, and the horizontal axis is vertical velocity. The robot moves around the origin in a counterclockwise direction. In the top two quadrants the robot is in the air, and in the bottom two quadrants the robot is on the ground. Thus, the horizontal axis is a discontinuity of the robot dynamics, and trajectory segments end and often begin at the discontinuity. We see that while the robot is in the air it cannot change how much energy it has (how high it goes or how fast it is going when it hits the ground), as the trajectories end with the same pattern they began with. When the robot is on the ground it thrusts with its leg to "focus" the trajectories so the set of touchdown positions is mapped to a smaller set of takeoff positions. This funneling effect is characteristic of controllers for periodic tasks, and how fast the funnel becomes narrow is controlled by the size of the penalty on $\mathbf{u}$ usage (Figure 2).

## 2.1  How are trajectory start points chosen?

In our approach trajectories are refined towards optimality given their fixed starting points. However, an initial trajectory must first be created. For regulator tasks, the trajectory is trivial and simply starts and ends at the known goal point. For tasks with a point goal, trajectories can be extended backwards away from the goal [3]. For periodic tasks, crude trajectories must be created using some other approach before this approach can refine

them.

We have used several methods to provide initial trajectories. Manually designed controllers sometimes work. In learning from demonstration a teacher provides initial trajectories [6]. In policy optimization (aka "policy search") a parameterized policy is optimized [9].

Once a set of initial task trajectories are available, the following four methods are used to generate trajectories in new parts of state space. We use all of these methods simultaneously, and locally optimize each of the trajectories produced. The best trajectory of the set is then stored and the other trajectories are discarded. 1) Use the global policy generated by policy optimization, if available. 2) Use the local policy from the nearest point with the same type of dynamics. 3) Use the local value function estimate (and derivatives) from the nearest point with the same type of dynamics. and 4) Use the policy from the nearest trajectory, where the nearest trajectory is selected at the beginning of the forward sweep and kept the same throughout the sweep. Note that methods 2 and 3 can change which stored trajectories they take points from on each time step, while method 4 uses a policy from a single neighboring trajectory.

## 3   Control of a walking robot

As another example we will describe the search for a policy for walking of a simple planar biped robot that walks along a bar. The simulated robot has two legs and a torque motor between the legs. Instead of revolute or telescoping knees, the robot can grab the bar with its foot as its leg swings past it. This is a model of a robot that walks along the trusses of a large structure such as a bridge, much as a monkey brachiates with its arms. This simple model has also been used in studies of robot passive dynamic walking [10].

This arrangement means the robot has a five dimensional state space: left leg angle ($\theta_l$), right leg angle ($\theta_r$), left leg angular velocity ($\dot{\theta}_l$), right leg angular velocity ($\dot{\theta}_r$), and stance foot location. A simple policy is used to determine when to grab the bar (at the end of a step when the swing foot passes the bar going downwards). The variable to be controlled is the torque $\tau$ at the hip.

The criterion we used is quite complex. We are a long way from specifying an abstract or vague criterion such as "cover a fixed distance with minimum fuel or battery usage" or "maximize the amount of your genes in future gene pools" and successfully finding an optimal or reasonable policy. At this stage we need to include several "shaping" terms in the criterion, that reward keeping the hips at the right altitude with minimal vertical velocity, keeping the leg amplitude within reason, maintaining a symmetric gait, and maintaining the desired hip forward velocity:

$$cost = w_y(y-1)^2 + w_{\dot{y}}\dot{y}^2 + w_l leg_l^2 + w_l leg_r^2 + w_{lr} leg_{lr} + w_{\dot{x}}(\dot{x} - \dot{x}_d)^2 + \tau^2 \quad (6)$$

where the $w_?$ are weighting factors and are $w_y = 100$, $w_{\dot{y}} = 100$. $w_l = 100$, $w_{lr} = 100000$, and $w_{\dot{x}} = 100$. The leg length is 1 meter (hence the 1 in $w_y(y-1)^2$). The desired leg velocity $\dot{x}_d = 0.4m/s$. $leg_?$ provides a measure of how far the left or right leg has gone past its limits $\pm 0.1 radians$ in the forward or backward direction. $leg_{lr}$ is the product of the leg angles if the legs are both forward or both rearward, and zero otherwise. $(x, y)$ is the hip location. The integration and control time steps are 1 millisecond each. The dynamics of this walker are simulated using a commercial package, SDFAST.

Initial trajectories were generated by optimizing the coefficients of a linear policy. When the left leg was in stance:

$$\tau = \alpha_0 + \alpha_1\theta_{lr} + \alpha_2\theta_l + \alpha_3 y + \alpha_4\dot{\theta}_{lr} + \alpha_5\dot{\theta}_l + \alpha_6\dot{x} + \alpha_7\dot{y} \quad (7)$$

where $\theta_{lr}$ is the angle between the legs. When the right leg was in stance the same policy was used with the appropriate signs negated.

### 3.1 Results

The trajectory-based approach was able to find a cheaper and more robust policy than the parametric policy-optimization approach. This is not surprising given the flexible and expandable representational capacity of an adaptive non-parametric representation, but it does provide some indication that our update algorithms can usefully harness the additional representation power.

**Cost:** For example, after training the parametric policy, we measured the undiscounted cost over 1 second (roughly one step of each leg) starting in a state along the lowest cost cyclic trajectory. The cost for the optimized parametric policy was 4316. The corresponding cost for the trajectory-based approach starting from the same state was 3502.

**Robustness:** We did a simple assessment of robustness by adding offsets to the same starting state until the optimized linear policy failed. The offsets were in terms of the stance leg and the angle between the legs, and the corresponding angular velocities. The maximum offsets for the linearized optimized parametric policy are $-0.02 <= \theta_l <= 0.06$, $-0.45 <= \dot{\theta}_l <= 0.1$, $-0.2 <= \theta_{lr} <= 0.03$, and $-0.78 <= \dot{\theta}_{lr} <= 0.2$. We did a similar test for the trajectory approach. In each direction the maximum offset the trajectory-based approach was able to handle was equal to or greater than the parametric policy-based approach, extending the range most in the cases of $\theta_{lr} <= 0.1$ and $\dot{\theta}_{lr} <= 1.0$. This is not surprising, since the trajectory-based controller uses the parametric policy as one of the ways to initially generate candidate trajectories for optimization. In cases where the trajectory-based approach is not able to generate an appropriate trajectory, the system will generate a series of trajectories with start points moving from regions it knows how to handle towards the desired start point. Thus, we have not yet discovered situations that are physically possible to recover that the trajectory-based approach cannot handle if it is allowed as much computation time as it needs.

**Interference:** To demonstrate interference in the parametric policy approach, we optimized its performance from a distribution of starting states. These states were the original state, and states with positive offsets. The new cost for the original starting position was 14,747, compared to 4316 before retraining. The trajectory approach has the same cost as before, 3502.

## 4 Robustness to modeling error and imperfect sensing

So far we have addressed robustness in terms of the range of initial states that can be handled. Another form of robustness is robustness to modeling error (changes in masses, friction, and other model parameters) and imperfect sensing, so that the controller does not know exactly what state the robot is in. Since simulations are used to optimize policies, it is relatively easy to include simulations with different model parameters and sensor noise in the training and optimize for a robust parametric controller in policy shaping. How does the trajectory-based approach achieve comparable robustness?

We have developed two approaches, a probabilistic approach with maintains distributional information about unknown states and parameters, and a game-based or minimax approach. The probabilistic approach supports actions by the controller to actively minimize uncertainty as well as achieve goals, which is known as dual control. The game-based approach does not reduce uncertainty with experience, and is somewhat paranoid, assuming the world is populated by evil spirits which choose the worst possible disturbance at each time step for the controller. This results in robust, but often overly conservative policies.

In the probabilistic case, the state is augmented with any unknown parameters such as masses of parts or friction coefficients, and the covariance of all the original elements of

the state as well as the added parameters. An extended Kalman filter is constructed as the new dynamics equation, predicting the new estimates of the means and covariances given the control signals to the system. The one step cost function is restated in terms of the augmented state. The value function is now a function of the augmented state, including covariances of the original state vector elements. These covariances interact with the curvature of the value function, causing additional cost in areas of the value function that have high curvature or second derivatives. Thus the system is rewarded when it moves to areas of the value function that are planar, and uncertainty has no effect on the expected cost. The system is also rewarded when it learns, which reduces the covariances of the estimates, so the system may choose actions that move away from a goal but reduce uncertainty. This probabilistic approach does dramatically increase the dimensionality of the state vector and thus the value function, but in the context of only a quadratic cost on dimensionality this is not as fatal is it would seem.

A less expensive approach is to use a game-based uncertainty model with minimax optimization. In this case, we assume an opponent can pick a disturbance to maximally increase our cost. This is closely related to robust nonlinear controller design techniques based on the idea of $H_\infty$ control [11, 12] and risk sensitive control [13, 14]. We augment the dynamics equation with a disturbance term: $\mathbf{x}_{k+1} = \mathbf{f}(\mathbf{x}, \mathbf{u}, \mathbf{v}) = \mathbf{f}_{original}(\mathbf{x}, \mathbf{u}) + \mathbf{v}$ where $\mathbf{v}$ is a vector of disturbance inputs. To limit the size of the disturbances, we include the disturbance magnitude in a modified one step cost function with a negative sign. The opponent who controls the disturbance wants to increase our cost, so this new term gives an incentive to the opponent to choose the worse direction for the disturbance, and a disturbance magnitude that gives the highest ratio of increased cost to disturbance size: $L(\mathbf{x}, \mathbf{u}, \mathbf{v}) = L_{original}(\mathbf{x}, \mathbf{u}) - \mathbf{v}^T \mathbf{M} \mathbf{v}$. Initially, $\mathbf{M}$ is set to globally approximate the uncertainty of the model. Ultimately, $\mathbf{M}$ should vary with the local confidence in the model. Highly practiced movements or portions of movements should have high $\mathbf{M}$, and new movements should have lower $\mathbf{M}$. The optimal action is now given by Isaacs' equation: $(\mathbf{v}, \mathbf{u}) = \arg\max_{\mathbf{v}} \min_{\mathbf{u}} (L(\mathbf{x}, \mathbf{u}, \mathbf{v}) + \lambda V_{k+1}(\mathbf{f}(\mathbf{x}, \mathbf{u}, \mathbf{v})))$. How we solve Isaacs' equation and an application of this method are described in the companion paper [15].

## 5   How to cover a volume of state space

In tasks with a goal or point attractor, [3] showed that certain key trajectories can be grown backwards from the goal in order to approximate the value function. In the case of a sparse use of trajectories to cover a space, the cost of the approach is dominated by the $d^2$ costs of updating second derivative matrices, and thus the cost of the trajectory-based approach increases quadratically as the dimensionality increases.

However, for periodic tasks the approach of growing trajectories backwards from the goal cannot be used, as there is no goal point or set. In this case the trajectories that form the optimal cycle can be used as key trajectories, with each point along them supplying a local linear policy and local quadratic value function. These key trajectories can be computed using any optimization method, and then the corresponding policy and value function estimates along the trajectory computed using the update rules given here.

It is important to point out that optimal trajectories need only be placed densely enough to separate regions which have different local optima. The trajectories used in the representation usually follow local valleys of the value function. Also, we have found that natural behavior often lies entirely on a low-dimensional manifold embedded in a high dimensional space. Using these trajectories and creating new trajectories as task demands require it, we expect to be able to handle a range of natural tasks.

# 6   Contributions

In order to accommodate periodic tasks, this paper has discussed how to incorporate discount factors into the trajectory-based approach, how to handle discontinuities in the dynamics (and equivalently, criteria and constraints), and how to find key trajectories for a sparse trajectory-based approach. The trajectory-based approach requires less design skill from humans since it doesn't need a "good" policy parameterization, produces cheaper and more robust policies, which do not suffer from interference.

# References

[1] Richard S. Sutton. Integrated architectures for learning , planning and reacting based on approximating dynamic programming. In *Proceedings 7th International Conference on Machine Learning.*, 1990.

[2] C. Atkeson and J. Santamaria. A comparison of direct and model-based reinforcement learning, 1997.

[3] Christopher G. Atkeson. Using local trajectory optimizers to speed up global optimization in dynamic programming. In Jack D. Cowan, Gerald Tesauro, and Joshua Alspector, editors, *Advances in Neural Information Processing Systems*, volume 6, pages 663–670. Morgan Kaufmann Publishers, Inc., 1994.

[4] P. Dyer and S. R. McReynolds. *The Computation and Theory of Optimal Control*. Academic Press, New York, NY, 1970.

[5] D. H. Jacobson and D. Q. Mayne. *Differential Dynamic Programming*. Elsevier, New York, NY, 1970.

[6] Christopher G. Atkeson and Stefan Schaal. Robot learning from demonstration. In *Proc. 14th International Conference on Machine Learning*, pages 12–20. Morgan Kaufmann, 1997.

[7] C. G. Atkeson, A. W. Moore, and S. Schaal. Locally weighted learning. *Artificial Intelligence Review*, 11:11–73, 1997.

[8] W. Schwind and D. Koditschek. Control of forward velocity for a simplified planar hopping robot. In *International Conference on Robotics and Automation*, volume 1, pages 691–6, 1995.

[9] J. Andrew Bagnell and Jeff Schneider. Autonomous helicopter control using reinforcement learning policy search methods. In *International Conference on Robotics and Automation*, 2001.

[10] M. Garcia, A. Chatterjee, and A. Ruina. Efficiency, speed, and scaling of two-dimensional passive-dynamic walking. *Dynamics and Stability of Systems*, 15(2):75–99, 2000.

[11] K. Zhou, J. C. Doyle, and K. Glover. *Robust Optimal Control*. PRENTICE HALL, New Jersey, 1996.

[12] J. Morimoto and K. Doya. Robust Reinforcement Learning. In Todd K. Leen, Thomas G. Dietterich, and Volker Tresp, editors, *Advances in Neural Information Processing Systems 13*, pages 1061–1067. MIT Press, Cambridge, MA, 2001.

[13] R. Neuneier and O. Mihatsch. Risk Sensitive Reinforcement Learning. In M. S. Kearns, S. A. Solla, and D. A. Cohn, editors, *Advances in Neural Information Processing Systems 11*, pages 1031–1037. MIT Press, Cambridge, MA, USA, 1998.

[14] S. P. Coraluppi and S. I. Marcus. Risk-Sensitive and Minmax Control of Discrete-Time Finite-State Markov Decision Processes. *Automatica*, 35:301–309, 1999.

[15] J. Morimoto and C. Atkeson. Minimax differential dynamic programming: An application to robust biped walking. In *Advances in Neural Information Processing Systems 15*. MIT Press, Cambridge, MA, 2002.
